# Analysis of distributed representation of constituent structure in connectionist systems

Paul Smolensky

Department of Computer Science, University of Colorado, Boulder, CO 80309-0430

## Abstract

A general method, the *tensor product representation*, is described for the distributed representation of value/variable bindings. The method allows the fully distributed representation of symbolic structures: the roles in the structures, as well as the fillers for those roles, can be arbitrarily non-local. Fully and partially localized special cases reduce to existing cases of connectionist representations of structured data; the tensor product representation generalizes these and the few existing examples of fully distributed representations of structures. The representation saturates gracefully as larger structures are represented; it permits recursive construction of complex representations from simpler ones; it respects the independence of the capacities to generate and maintain multiple bindings in parallel; it extends naturally to continuous structures and continuous representational patterns; it permits values to also serve as variables; it enables analysis of the interference of symbolic structures stored in associative memories; and it leads to characterization of optimal distributed representations of roles and a recirculation algorithm for learning them.

## Introduction

Any model of complex information processing in networks of simple processors must solve the problem of representing complex structures over network elements. Connectionist models of realistic natural language processing, for example, must employ computationally adequate representations of complex sentences. Many connectionists feel that to develop connectionist systems with the computational power required by complex tasks, distributed representations must be used: an individual processing unit must participate in the representation of multiple items, and each item must be represented as a pattern of activity of multiple processors. Connectionist models have used more or less distributed representations of more or less complex structures, but little if any general analysis of the problem of distributed representation of complex information has been carried out. This paper reports results of an analysis of a general method called the *tensor product representation*.

The language-based formalisms traditional in AI permit the construction of arbitrarily complex structures by piecing together constituents. The tensor product representation is a connectionist method of combining representations of constituents into representations of complex structures. If the constituents that are combined have distributed representations, completely distributed representations of complex structures can result: each part of the network is responsible for representing multiple constituents in the structure, and each constituent is represented over multiple units. The tensor product representation is a general technique, of which the few existing examples of fully distributed representations of structures are particular cases.

The tensor product representation rests on identifying natural counterparts within connectionist computation of certain fundamental elements of symbolic computation. In the present analysis, the problem of distributed representation of symbolic structures is characterized as the problem of taking complex structures with certain relations to their constituent symbols and mapping them into activity vectors—patterns of activation—with corresponding relations to the activity vectors representing their constituents. Central to the analysis is identifying a connectionist counterpart of *variable binding*: a method for binding together a distributed representation of a variable and a distributed representation of a value into a distributed representation of a variable/value binding—a representation which can co-exist on exactly the same network units with representations of other variable/value bindings, with

limited confusion of which variables are bound to which values.

In summary, the analysis of the tensor product representation

(1) provides a general technique for constructing fully distributed representations of arbitrarily complex structures;

(2) clarifies existing representations found in particular models by showing what particular design decisions they embody;

(3) allows the proof of a number of general computational properties of the representation;

(4) identifies natural counterparts within connectionist computation of elements of symbolic computation, in particular, variable binding.

The recent emergence to prominence of the connectionist approach to AI raises the question of the relation between the nonsymbolic computation occurring in connectionist systems and the symbolic computation traditional in AI. The research reported here is part of an attempt to marry the two types of computation, to develop for AI a form of computation that adds crucial aspects of the power of symbolic computation to the power of connectionist computation: massively parallel soft constraint satisfaction. One way to marry these approaches is to implement serial symbol manipulation in a connectionist system[1,2]. The research described here takes a different tack. In a massively parallel system the processing of symbolic structures—for example, representations of parsed sentences—need not be limited to a series of elementary manipulations: indeed one would expect the processing to involve massively parallel soft constraint satisfaction. But in order for such processing to occur, a satisfactory answer must be found for the question: *How can symbolic structures, or structured data in general, be naturally represented in connectionist systems?* The difficulty here turns on one of the most fundamental problems for relating symbolic and connectionist computation: *How can variable binding be naturally performed in connectionist systems?*

This paper provides an overview of a lengthy analysis reported elsewhere[3] of a general connectionist method for variable binding and an associated method for representing structured data. The purpose of this paper is to introduce the basic idea of the method and survey some of the results; the reader is referred to the full report for precise definitions and theorems, more extended examples, and proofs.

## The problem

Suppose we want to represent a simple structured object, say a sequence of elements, in a connectionist system. The simplest method, which has been used in many models, is to dedicate a network processing unit to each possible element in each possible position[4-9]. This is a *purely local representation*. One way of looking at the purely local representation is that the binding of constituents to the variables representing their positions is achieved by dedicating a separate unit to every possible binding, and then by activating the appropriate individual units.

Purely local representations of this sort have some advantages[10], but they have a number of serious problems. Three immediately relevant problems are these:

(1) The number of units needed is *#elements * #positions*; most of these processors are inactive and doing no work at any given time.

(2) The number of positions in the structures that can be represented has a fixed, rigid upper limit.

(3) If there is a notion of similar elements, the representation does not take advantage of this: similar sequences do not have similar representations.

The technique of *distributed representation* is a well-known way of coping with the first and third problems[11-14]. If elements are represented as *patterns of activity* over a population of processing units, and if each unit can participate in the representation of many elements, then the number of elements that can be represented is much greater than the number of units, and similar elements can be represented by similar patterns, greatly enhancing the power of the network to learn and take advantage of generalizations.

Distributed representations of elements in structures (like sequences) have been successfully used in many models[1,4,5,15–18]. For each position in the structure, a *pool of units* is dedicated. The element occurring in that position is represented by a pattern of activity over the units in the pool.

Note that this technique goes only part of the way towards a truly distributed representation of the entire structure. While the *values* of the variables defining the roles in the structure are represented by distributed patterns instead of dedicated units, the *variables themselves* are represented by localized, dedicated pools. For this reason I will call this type of representation *semi-local*.

Because the representation of variables is still local, semi-local representations retain the second of the problems of purely local representations listed above. While the generic behavior of connectionist systems is to gradually overload as they attempt to hold more and more information, with dedicated pools representing role variables in structures, there is no loading at all until the pools are exhausted—and then there is complete saturation. The pools are essentially *registers*, and the representation of the structure as a whole has more of the characteristics of von Neumann storage than connectionist representation. A fully distributed connectionist representation of structured data would saturate gracefully.

Because the representation of variables in semi-local representations is local, semi-local representations also retain part of the third problem of purely local representations. Similar elements have similar representations *only if they occupy exactly the same role in the structure*. A notion of similarity of roles cannot be incorporated in the semi-local representation.

## Tensor product binding

There is a way of viewing both the local and semi-local representations of structures that makes a generalization to fully distributed representations immediately apparent. Consider the following structure: strings of length no more than four letters. Fig. 1 shows a purely local representation and Fig. 2 shows a semi-local representation (both of which appeared in the letter-perception model of McClelland and Rumelhart[4,5]). In each case, the variable binding has been viewed in the same way. On the left edge is a set of imagined units which can represent an element in the structure—a filler of a role; these are the *filler units*. On the bottom edge is a set of imagined units which can represent a role: these are the *role units*. The remaining units are the ones really used to represent the structure: the *binding units*. They are arranged so that there is one for each pair of filler and role units.

In the purely local case, both the filler and the role are represented by a "pattern of activity" localized to a single unit. In the semi-local case, the filler is represented by a distributed pattern of activity but the role is still represented by a localized pattern. In either case, the binding of the filler to the role is achieved by a simple product operation: the activity of each binding unit is the product of the activities of the associated filler and role unit. In the vocabulary of matrix algebra, the activity representing the value/variable binding forms a matrix which is the *outer product* of the activity vector representing the value and the activity vector representing the variable. In the terminology of vector spaces, *the value/variable binding vector is the tensor product of the value vector and the variable vector*. This is what I refer to as the *tensor product representation* for variable bindings.

Since the activity vectors for roles in Figs. 1 and 2 consist of all zeroes except for a single activity of 1, the tensor product operation is utterly trivial. The local and semi-local cases are trivial special cases of a general binding procedure capable of producing completely distributed representations. Fig. 3 shows a distributed case designed for visual transparency. Imagine we are representing speech data, and have a sequence of values for the energy in a particular formant at successive times. In Fig. 3, distributed patterns are used to represent both the energy value and the variable to which it is bound: the position in time. The particular binding shown is of an energy value 2 (on a scale of 1–4) to the time 4. The peaks in the patterns indicate the value and variable being represented.

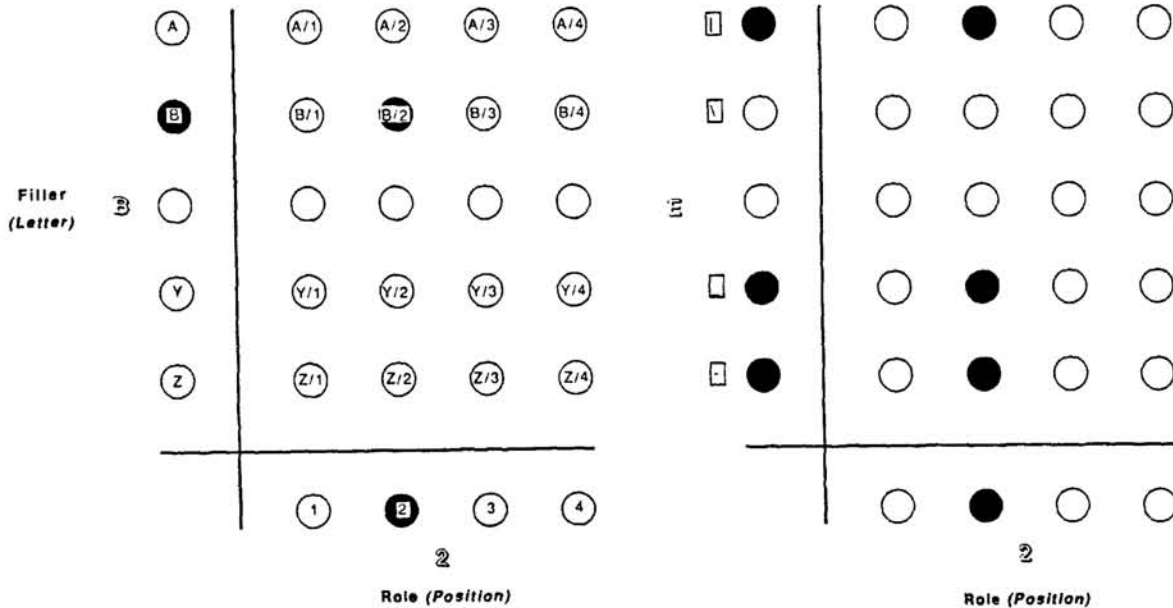

**Fig. 1.** Purely local representation of strings.   **Fig. 2.** Semi-local representation of strings.

If the patterns representing the value and variable being bound together are not as simple as those used in Fig. 3, the tensor product pattern representing the binding will not of course be particularly visually informative. Such would be the case if the patterns for the fillers and roles in a structure were defined with respect to a set of filler and role *features*: such distributed bindings have been used effectively by McClelland and Kawamoto[18] and by Derthick[19,20]. The extreme mathematical simplicity of the tensor product operation makes feasible an analysis of the general, fully distributed case.

Each binding unit in the tensor product representation corresponds to a pair of imaginary role and filler units. A binding unit can be readily interpreted semantically if its corresponding filler and role units can. The activity of the binding unit indicates that in the structure being represented an element is present which possesses the feature indicated by the corresponding filler unit *and* which occupies a role in the structure which possesses the feature indicated by the corresponding role unit. The binding unit thus detects a *conjunction* of a pair of filler and role features. (Higher-order conjunctions will arise later.)

A structure consists of multiple filler/role bindings. So far we have only discussed the representation of a single binding. In the purely local and semi-local cases, there are separate pools for different roles, and it is obvious how to combine bindings: simultaneously represent them in the separate pools. In the case of a fully distributed tensor product binding (eg., Fig. 3), each single binding is a pattern of activity that extends across the entire set of binding units. The simplest possibility for combining these patterns is simply to *add them up*; that is, to superimpose all the bindings on top of each other. In the special cases of purely local and semi-local representations, this procedure reduces trivially to simultaneously representing the individual fillers in the separate pools.

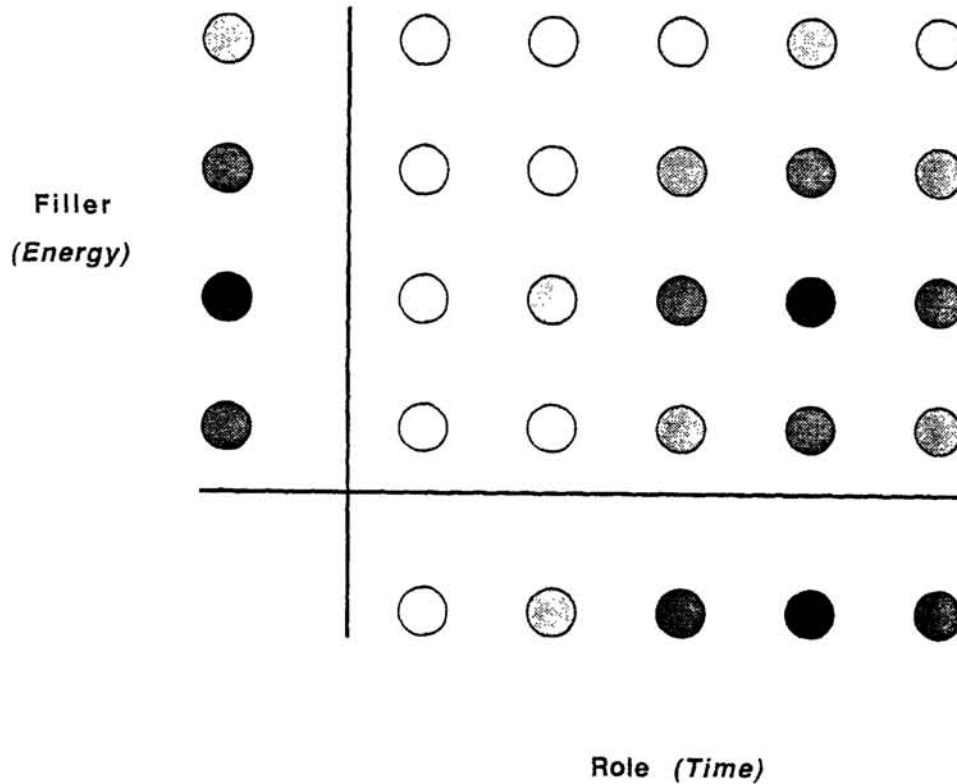

Filler

(Energy)

Role  (Time)

**Fig. 3.** A visually transparent fully distributed tensor product representation.

The process of superimposing the separate bindings produces a representation of structures with the usual connectionist properties. If the patterns representing the roles are not too similar, the separate bindings can all be kept straight. It is not necessary for the role patterns to be non-overlapping, as they are in the purely local and semi-local cases; it is sufficient that the patterns be linearly independent. Then there is a simple operation that will correctly extract the filler for any role from the representation of the structure as a whole. If the patterns are not just linearly independent, but are also orthogonal, this operation becomes quite direct; we will get to it shortly. For now, the point is that simply superimposing the separate bindings is sufficient. If the role patterns are not too similar, the separate bindings do not interfere. The representation gracefully saturates if more and more roles are filled, since the role patterns being used lose their distinctness once their number approaches that of the role units.

Thus problem (2) listed above, shared by purely local and semi-local representations, is at last removed in fully distributed tensor product representations: they do not accomodate structures only up to a certain rigid limit, beyond which they are completely saturated; rather, they saturate gracefully. The third problem is also fully addressed, as similar roles can be represented by similar patterns in the tensor product representation and then generalizations both across similar fillers and across similar roles can be learned and exploited.

The definition of the tensor product representation of structured data can be summed up as follows:

(a)  Let a set $S$ of structured objects be given a *role decomposition*: a set of fillers, $F$, a set of roles $R$, and for each object $s$ a corresponding set of bindings $\beta(s) = \{f/r : f \text{ fills role } r \text{ in } s\}$.

(b)  Let a connectionist representation of the fillers $F$ be given; $f$ is represented by the activity vector **f**.

(c)  Let a connectionist representation of the roles $R$ be given; $r$ is represented by the activity

vector **r**.

(d) Then the corresponding tensor product representation of $s$ is $\sum\limits_{f/r \in \beta(s)} f \otimes r$ (where $\otimes$ denotes the tensor product operation).

In the next section I will discuss a model using a fully distributed tensor product representation, which will require a brief consideration of role decompositions. I will then go on to summarize general properties of the tensor product representation.

## Role decompositions

The most obvious role decompositions are *positional decompositions* that involve fixed position slots within a structure of pre-determined form. In the case of a string, such a role would be the $i^{th}$ position in the string; this was the decomposition used in the examples of Figs. 1 through 3. Another example comes from McClelland and Kawamoto's model[18] for learning to assign case roles. They considered sentences of the form *The $N_1$ V the $N_2$ with the $N_3$*; the four roles were the slots for the three nouns and the verb.

A less obvious but sometimes quite powerful role decomposition involves not fixed positions of elements but rather their *local context*. As an example, in the case of strings of letters, such roles might be $r_{x\_y} = $ *is preceded by x and followed by y*, for various letters $x$ and $y$.

Such a local context decomposition was used to considerable advantage by Rumelhart and McClelland in their model of learning the morphophonology of the English past tense[21]. Their structures were strings of phonetic segments, and the context decomposition was well-suited for the task because the generalizations the model needed to learn involved the transformations undergone by phonemes occurring in different local contexts.

Rumelhart and McClelland's representation of phonetic strings is an example of a fully distributed tensor product representation. The fillers were phonetic segments, which were represented by a pattern of phonetic features, and the roles were nearest-neighbor phonetic contexts, which were also represented as distributed patterns. The distributed representation of the roles was in fact itself a tensor product representation: the roles themselves have a constituent structure which can be further broken down through another role decomposition. The roles are indexed by a left and right neighbor; in essence, a string of two phonetic segments. This string too can be decomposed by a context decomposition; the filler can be taken to be the left neighbor, and the role can be indexed by the right neighbor. Thus the vowel $[i]$ in the word *week* is bound to the role $r_{w\_k}$, and this role is in turn a binding of the filler $[w]$ in the sub-role $r'_{\_k}$. The pattern for $[i]$ is a vector **i** of phonetic features; the pattern for $[w]$ is another such vector of features **w**, and the pattern for the sub-role $r'_{\_k}$ is a third vector **k** consisting of the phonetic features of $[k]$. The binding for the $[i]$ in *week* is thus $\mathbf{i} \otimes (\mathbf{w} \otimes \mathbf{k})$. Each unit in the representation represents a *third-order* conjunction of a phonetic feature for a central segment together with two phonetic features for its left and right neighbors. [To get precisely the representation used by Rumelhart and McClelland, we have to take this tensor product representation of the roles (eg. $r_{w\_k}$) and throw out a number of the binding units generated in this further decomposition; only certain combinations of features of the left and right neighbors were used. The distributed representation of letter triples used by Touretzky and Hinton[1] can be viewed as a similar third-order tensor product derived from nested context decompositions, with some binding units thrown away—in fact, all binding units off the main diagonal were discarded.]

This example illustrates how role decompositions can be iterated, leading to iterated tensor product representations. Whenever the fillers or roles of one decomposition are structured objects, they can themselves be further reduced by another role decomposition.

It is often useful to consider recursive role decompositions in which the fillers are the same type of object as the original structure. It is clear from the above definition that such a decomposition cannot be used to generate a tensor product representation. Nonetheless, recursive role decompositions *can* be used to relate the tensor product representation of complex structures to the tensor product representations of simpler structures. For example, consider Lisp binary tree structures built from a set $A$ of atoms. A non-recursive decomposition uses $A$ as the set of fillers, with each role being the occupation of a certain position in the tree by an atom. From this decomposition a tensor product representation can be constructed. Then it can be seen that the operations *car*, *cdr*, and *cons* correspond to certain linear operators **car**, **cdr**, and **cons** in the vector space of activation vectors. Just as complex S-expressions can be constructed from atoms using *cons*, so their connectionist representations can be constructed from the simple representation of atoms by the application of **cons**. (This serial "construction" of the complex representation from the simpler ones is done by *the analyst*, not necessarily by the network; **cons** is a static, descriptive, mathematical operator—not necessarily a transformation to be carried out by a network.)

## Binding and unbinding in connectionist networks

So far, the operation of binding a value to a variable has been described mathematically and pictured in Figs. 1–3 in terms of "imagined" filler units and role units. Of course, the binding operation can actually be performed in a network if the filler and role units are really there. Fig. 4 shows one way this can be done. The triangular junctions are Hinton's multiplicative connections[22]: the incoming activities from the role and filler units are multiplied at the junction and passed on to the binding unit.

**Binding Units**

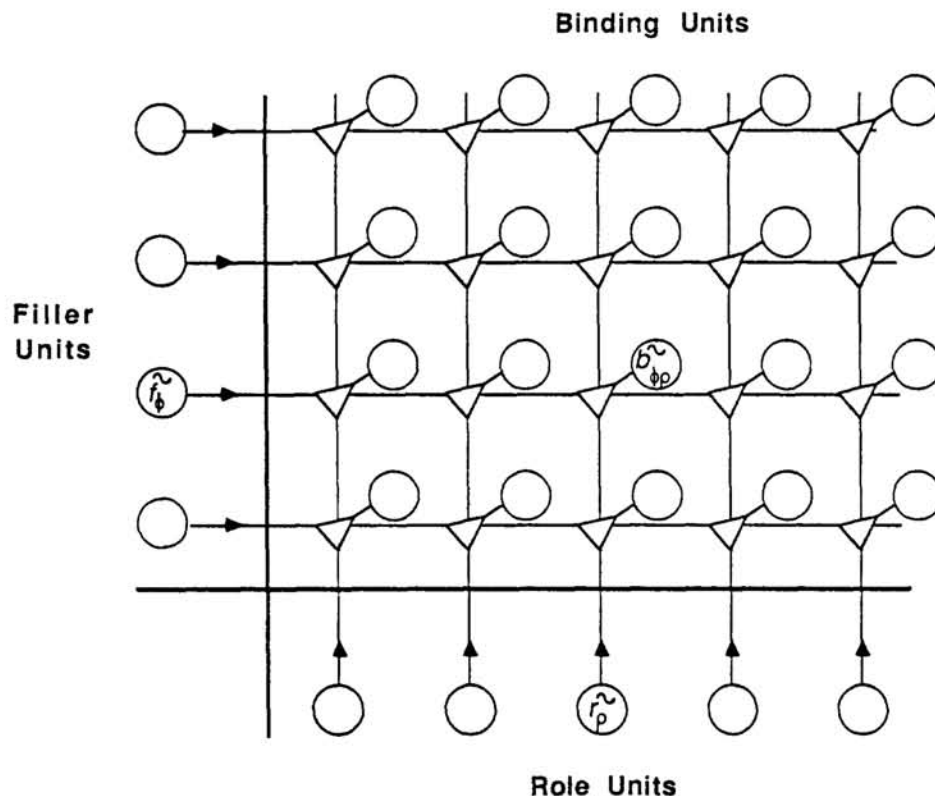

**Filler Units**

**Role Units**

**Fig. 4.** A network for tensor product binding and unbinding.

"Unbinding" can also be performed by the network of Fig. 4. Suppose the tensor product representation of a structure is present in the binding units, and we want to extract the filler for a particular role. As mentioned above, this can be done accurately if the role patterns are linearly independent (and if each role is bound to only one filler). It can be shown that in this case, for each role there is a pattern of activity which, if set up on the role units, will lead to a pattern on the filler units that represents the corresponding filler. (If the role vectors are orthogonal, this pattern is the same as the role pattern.) As in Hinton's model[20], it is assumed here that the triangular junctions work in all directions, so that now they take the product of activity coming in from the binding and role units and pass it on to the filler units, which sum all incoming activity.

The network of Fig. 4 can bind one value/variable pair at a time. In order to build up the representation of an entire structure, the binding units would have to accumulate activity over an extended period of time during which all the individual bindings would be performed serially. Multiple bindings could occur in parallel if part of the apparatus of Fig. 4 were duplicated: this requires several copies of the sets of filler and role units, paired up with triangular junctions, all feeding into a single set of binding units.

Notice that in this scheme there are two *independent* capacities for parallel binding: the capacity to *generate* bindings in parallel, and the capacity to *maintain* bindings simultaneously. The former is determined by the degree of duplication of the filler/role unit sets (in Fig. 4, for example, the parallel generation capacity is 1). The parallel maintenance capacity is determined by the number of possible linearly independent role patterns, i.e. the number of role units in each set. It is logical that these two capacities should be independent, and in the case of the human visual and linguistic systems it seems that our maintenance capacity far exceeds our generation capacity[21]. Note that in purely local and semi-local representations, there is a separate pool of units dedicated to the representation of each role, so there is a tendency to suppose that the two capacities are equal. As long as a connectionist model deals with structures (like four-letter words) that are so small that the number of bindings involved is within the human parallel generation capacity, there is no harm done. But when connectionist models address the human representation of large structures (like entire scenes or discourses), it will be important to be able to maintain a large number of bindings even though the number that can be generated in parallel is much smaller.

## Further properties and extensions

*Continuous structures.* It can be argued that underlying the connectionist approach is a fundamentally continuous formalization of computation[13]. This would suggest that a natural connectionist representation of structure would apply at least as well to continuous structures as to discrete ones. It is therefore of some interest that the tensor product representation applies equally well to structures characterized by a continuum of roles: a "string" extending through continuous time, for example, as in continuous speech. In place of a sum over a discrete set of bindings, $\sum_i \mathbf{f}_i \otimes \mathbf{r}_i$ we have an integral over a continuum of bindings: $\int_t \mathbf{f}(t) \otimes \mathbf{r}(t) \, dt$ This goes over exactly to the discrete case if the fillers are discrete step-functions of time.

*Continuous patterns.* There is a second sense in which the tensor product representation extends naturally to the continuum. If the patterns representing fillers and/or roles are continuous curves rather than discrete sets of activities, the tensor product operation is still well-defined. (Imagine Fig. 3 with the filler and role patterns being continuous peaked curves instead of discrete approximations; the binding pattern is then a continuous peaked two-dimensional surface.) In this case, the vectors $\mathbf{f}$ and/or $\mathbf{r}$ are members of infinite-dimensional function spaces; regarding them as patterns of activity over a set of processors would require an infinite number of processors. While this might pose some problems for computer simulation, the case where $\mathbf{f}$ and/of $\mathbf{r}$ are functions rather than finite-dimensional vectors is not particularly problematic analytically. And if a problem with a continuum of roles is being considered, it may be desirable to assume a continuum of linearly independent role vectors: this requires considering infinite-dimensional representations.

*Values as variables.* Treating both values and variables symmetrically as done in the tensor product representation makes it possible for the same entity to simultaneously serve both as a value and as a variable. In symbolic computation it often happens that the value bound to one variable is itself a variable which in turn has a value bound to it. In a semi-local representation, where variables are localized pools of units and values are patterns of activity in these pools, it is difficult to see how the same entity can simultaneously serve as both value and variable. In the tensor product representation, both values and variables are patterns of activity, and whether a pattern is serving as a "variable" or "value"—or both—might be merely a matter of descriptive preference.

*Symbolic structures in associative memories.* The mathematical simplicity of the tensor product representation makes it possible to characterize conditions under which a set of symbolic structures can be stored in an associative memory without interference. These conditions involve an interesting mixture of the numerical character of the associative memory and the discrete character of the stored data.

*Learning optimal role patterns by recirculation.* While the use of distributed patterns to represent *constituents* in structures is well-known, the use of such patterns to represent *roles* in structures poses some new challenges. In some domains, features for roles are familiar or easy to imagine; eg., features of semantic roles in a case-frame semantics. But it is worth considering the problem of distributed role representations in domain-independent terms as well. The patterns used to represent roles determine how information about a structure's fillers will be coded, and these role patterns have an effect on how much information can subsequently be extracted from the representation by connectionist processing. The challenge of making the most information available for such future extraction can be posed as follows. Assume enough apparatus has been provided to do all the variable binding in parallel in a network like that of Fig. 4. Then we can dedicate a set of role units to each role; the pattern for each role can be set up once and for all in one set of role units. Since the activity of the role units provide multipliers for filler values at the triangular junctions, we can treat these fixed role patterns as weights on the lines from the filler units to the binding units. The problem of finding good role patterns now becomes the problem of finding good weights for encoding the fillers into the binding units.

Now suppose that a second set of connections is used to try to extract all the fillers from the representation of the structure in the binding units. Let the weights on this second set of connections be chosen to minimize the mean-squared differences between the extracted filler patterns and the actual original filler patterns. Let a set of role vectors be called *optimal* if this mean-squared error is as small as possible.

It turns out that optimal role vectors can be characterized fairly simply both algebraically and geometrically (with the help of results from Williams[24]). Furthermore, having imbedded the role vectors as weights in a connectionist net, it is possible for the network to learn optimal role vectors by a fairly simple learning algorithm. The algorithm is derived as a gradient descent in the mean-squared error, and is what G. E. Hinton and J. L. McClelland (unpublished communication) have called a *recirculation algorithm*: it works by circulating activity around a closed network loop and training on the difference between the activities at a given node on successive passes.

## Acknowledgements

This research has been supported by NSF grants IRI-8609599 and ECE-8617947, by the Sloan Foundation's computational neuroscience program, and by the Department of Computer Science and Institute of Cognitive Science at the University of Colorado at Boulder.

## References

1. D. S. Touretzky & G. E. Hinton. *Proceedings of the International Joint Conference on Artificial Intelligence*, 238-243 (1985).
2. D. S. Touretzky. *Proceedings of the 8th Conference of the Cognitive Science Society*, 522–530 (1986).
3. P. Smolensky. Technical Report CU–CS–355–87, Department of Computer Science, University of Colorado at Boulder (1987).
4. J. L. McClelland & D. E. Rumelhart. *Psychological Review* **88**, 375–407 (1981).
5. D. E. Rumelhart & J. L. McClelland. *Psychological Review* **89**, 60–94 (1982).
6. M. Fanty. Technical Report 174, Department of Computer Science, University of Rochester (1985).
7. J. A. Feldman. *The Behavioral and Brain Sciences* **8**, 265–289 (1985).
8. J. L. McClelland & J. L. Elman. In J. L. McClelland, D. E. Rumelhart, & the PDP Research Group, *Parallel distributed processing: Explorations in the microstructure of cognition. Vol. 2: Psychological and biological models.* Cambridge, MA: MIT Press/Bradford Books, 58–121 (1986).
9. T. J. Sejnowski & C. R. Rosenberg. *Complex Systems* **1**, 145–168 (1987).
10. J. A. Feldman. Technical Report 189, Department of Computer Science, University of Rochester (1986).
11. J. A. Anderson & G. E. Hinton. In G. E. Hinton and J. A. Anderson, Eds., *Parallel models of associative memory.* Hillsdale, NJ: Erlbaum, 9–48 (1981).
12. G. E. Hinton, J. L. McClelland, & D. E. Rumelhart. In D. E. Rumelhart, J. L. McClelland, & the PDP Research Group, *Parallel distributed processing: Explorations in the microstructure of cognition. Vol. 1: Foundations.* Cambridge, MA: MIT Press/Bradford Books, 77–109 (1986).
13. P. Smolensky. *The Behavioral and Brain Sciences* **11**(1) (in press).
14. P. Smolensky. In J. L. McClelland, D. E. Rumelhart, & the PDP Research Group, *Parallel distributed processing: Explorations in the microstructure of cognition. Vol. 2: Psychological and biological models.* Cambridge, MA: MIT Press/Bradford Books, 390–431 (1986).
15. G. E. Hinton. In Hinton, G.E. and Anderson, J.A., Eds., *Parallel models of associative memory.* Hillsdale, NJ: Erlbaum, 161–188 (1981).
16. M. S. Riley & P. Smolensky. *Proceedings of the Sixth Annual Conference of the Cognitive Science Society*, 286–292 (1984).
17. P. Smolensky. In D. E. Rumelhart, J. L. McClelland, & the PDP Research Group, *Parallel distributed processing: Explorations in the microstructure of cognition. Vol. 1: Foundations.* Cambridge, MA: MIT Press/Bradford Books, 194–281 (1986).
18. J. L. McClelland & A. H. Kawamoto. In J. L. McClelland, D. E. Rumelhart, & the PDP Research Group, *Parallel distributed processing: Explorations in the microstructure of cognition. Vol. 2: Psychological and biological models.* Cambridge, MA: MIT Press/Bradford Books, 272–326 (1986).
19. M. Derthick. *Proceedings of the National Conference on Artificial Intelligence*, 346–351 (1987).
20. M. Derthick. *Proceedings of the Annual Conference of the Cognitive Science Society*, 131–142 (1987).
21. D. E. Rumelhart & J. L. McClelland. In J. L. McClelland, D. E. Rumelhart, & the PDP Research Group, *Parallel distributed processing: Explorations in the microstructure of cognition. Vol. 2: Psychological and biological models.* Cambridge, MA: MIT Press/Bradford Books, 216–271 (1986)
22. G. E. Hinton. *Proceedings of the Seventh International Joint Conference on Artificial Intelligence*, 683–685 (1981).
23. A. M. Treisman & H. Schmidt. *Cognitive Psychology* **14**, 107–141 (1982).
24. R. J. Williams. Technical Report 8501, Institute of Cognitive Science, University of California, San Diego (1985).
